# Stability-Based Model Selection

**Tilman Lange, Mikio L. Braun, Volker Roth, Joachim M. Buhmann**
(lange,braunm,roth,jb)@cs.uni-bonn.de
Institute of Computer Science, Dept. III,
University of Bonn
Römerstraße 164, 53117 Bonn, Germany

## Abstract

Model selection is linked to *model assessment*, which is the problem of
comparing different models, or model parameters, for a specific learning
task. For supervised learning, the standard practical technique is cross-
validation, which is not applicable for semi-supervised and unsupervised
settings. In this paper, a new model assessment scheme is introduced
which is based on a notion of stability. The stability measure yields an
upper bound to cross-validation in the supervised case, but extends to
semi-supervised and unsupervised problems. In the experimental part,
the performance of the stability measure is studied for model order se-
lection in comparison to standard techniques in this area.

## 1 Introduction

One of the fundamental problems of learning theory is model assessment: Given a specific
data set, how can one practically measure the generalization performance of a model trained
to the data. In supervised learning, the standard technique is cross-validation. It consists in
using only a subset of the data for training, and then testing on the remaining data in order to
estimate the expected risk of the predictor. For semi-supervised and unsupervised learning,
there exist no standard techniques for estimating the generalization of an algorithm, since
there is no expected risk. Furthermore, in unsupervised learning, the problem of model
order selection arises, i.e. estimating the "correct" number of clusters. This number is part
of the input data for supervised and semi-supervised problems, but it is not available for
unsupervised problems.

We present a common point of view, which provides a unified framework for model as-
sessment in these seemingly unrelated areas of machine learning. The main idea is that an
algorithm generalizes well, if the solution on one data set has small disagreement with the
solution on another data set. This idea is independent of the amount of label information
which is supplied to the problem, and the challenge is to define disagreement in a mean-
ingful way, without relying on additional assumptions, e.g. mixture densities. The main
emphasis lies on developing model assessment procedures for semi-supervised and unsu-
pervised clustering, because a definitive answer to the question of model assessment has
not been given in these areas.

In section 3, we derive a stability measure for solutions to learning problems, which al-
lows us to characterize the generalization in terms of the stability of solutions on different
sets. For supervised learning, this stability measure is an upper bound to the 2-fold cross-

validation error, and can thus be understood as a natural extension of cross-validation to semi-supervised and unsupervised problems.

For experiments (section 4), we have chosen the model order selection problem in the unsupervised setting, which is one of the relevant areas of application as argued above. We compare the stability measure to other techniques from the literature.

## 2  Related Work

For supervised learning problems, several notions of stability have been introduced ([10], [3]). The focus of these works lies on deriving theoretical generalization bounds for supervised learning. In contrast, this work aims at developing practical procedures for model assessment, which are also applicable in semi- and unsupervised settings. Furthermore, the definition of stability developed in this paper does not build upon the cited works.

Several procedures have been proposed for inferring the number of clusters of which we name a few here. Tibshirani et al. [14] propose the *Gap Statistic* that is applicable to Euclidian data only. Given a clustering solution, the total sum of within-cluster dissimilarities is computed. This quantity computed on the original data is compared with the average over data which was uniformly sampled from a hyper-rectangle containing the original data. The $k$ which maximizes the *gap* between these two quantities is the estimated number of clusters. Recently, resampling-based approaches for model order selection have been proposed that perform model assessment in the spirit of cross validation. These approaches share the idea of *prediction strength* or *replicability* as a common trait. The methods exploit the idea that a clustering solution can be used to construct a predictor, in order to compute a solution for a second data set and to compare the computed and predicted class memberships for the second data set. In an early study, Breckenridge [4] investigated the usefulness of this approach (called *replication analysis* there) for the purpose of cluster validation. Although his work does not lead to a directly applicable procedure, in particular not for model order selection, his study suggests the usefulness of such an approach for the purpose of validation. Our method can be considered as a refinement of his approach. Fridlyand and Dudoit [6] propose a model order selection procedure, called *Clest*, that also builds upon Breckenridge's work. Their method employs the replication analysis idea by repeatedly splitting the available data into two parts. Free parameters of their method are the predictor, the measure of agreement between a computed and a predicted solution and a baseline distribution similar to the Gap Statistic. Because these three parameters largely influence the assessment, we consider their proposal more as a conceptual framework than as a concrete model order estimation procedure. In particular, the predictor can be chosen independent of the clustering algorithm which can lead to unreliable results (see section 3). For the experiments in section 4, we used a linear discriminant analysis classifier, the Fowlkes-Mellows index for solution comparison (c.f. [9, 6]) and the baseline distribution of the Gap Statistic. Tibshirani et al. [13] formulated a similar method (*Prediction Strength*) for inferring the number of clusters which is based on using nearest centroid predictors. Roughly, their measure of agreement quantifies the similarity of two clusters in the computed and in the predicted solution. For inferring a number of clusters, the least similar pair of clusters is taken into consideration. The estimated $\hat{k}$ is the largest $k$ for which the similarity is above some threshold value. Note that the similarity for $k = 1$ is always above this threshold.

## 3  The Stability Measure

We begin by introducing a stability measure for supervised learning. Then, the stability measure is generalized to semi-supervised and unsupervised settings. Necessary modifications for model order selection are discussed. Finally, a scheme for practical estimation of the stability is proposed.

**Stability and Supervised Learning**   The supervised learning problem is defined as follows. Let $\mathbf{Z} = (\mathbf{X}, \mathbf{Y}) = (X_1, Y_1), \ldots, (X_n, Y_n)$ be a sequence of random variables where $(X_i, Y_i)$ are drawn i.i.d. from some probability distribution $P_{X,Y}$. The $X_i \in \mathcal{X}$ are the objects and $Y_i \in \{1, \ldots, k\}$ are the labels. The task is to find a *labeling function* $g \colon \mathcal{X} \to \{1, \ldots, k\}$ which minimizes the expected risk, given by $R(g) := \int L(g(x), y) dP_{X,Y}$, using only a finite sample of data $\mathbf{Z}$ as input. Here $L$ is the so-called loss function. For classification, we take the 0-1-loss defined by $L(y, y') = \mathbf{1}\{y \neq y'\} = 1$ iff $y \neq y'$ and 0 else.

A measure of the stability of the labeling function learned is derived as follows. Note that for three labels $y$, $y'$ and $y''$, it holds that $\mathbf{1}\{y \neq y''\} \leq \mathbf{1}\{y \neq y'\} + \mathbf{1}\{y' \neq y''\}$, since $y \neq y''$ implies $y' \neq y$ or $y' \neq y''$. Now let $\mathbf{Z}$ and $\mathbf{Z}'$ be two data sets drawn independently from the same source, and denote the predictor $g$ trained on $\mathbf{Z}$ by $g_{\mathbf{Z}}$. Then, the *test risk* of $g_{\mathbf{Z}}$ can be bounded by introducing $g_{\mathbf{Z}'}(X_i')$:

$$R_{\mathbf{Z}'}(g_{\mathbf{Z}}) = \frac{1}{n} \sum_{i=1}^{n} \mathbf{1}\{g_{\mathbf{Z}}(X_i') \neq Y_i'\} \leq R_{\mathbf{Z}'}(g_{\mathbf{Z}'}) + \frac{1}{n} \sum_{i=1}^{n} \mathbf{1}\{g_{\mathbf{Z}}(X_i') \neq g_{\mathbf{Z}'}(X_i')\}. \quad (1)$$

We call the second term the *stability* of the predictor $g$ and denote its expectation by $S(g)$:

$$S(g) := E\Big[\frac{1}{n} \sum_{i=1}^{n} \mathbf{1}\{g_{\mathbf{Z}}(X_i') \neq g_{\mathbf{Z}'}(X_i')\}\Big]. \quad (2)$$

We call the value of $S(g)$ *stability cost* to stress the fact that $S = 0$ means perfect stability and large values of $S$ mean large instability. Taking expectations with respect to $\mathbf{Z}$ and $\mathbf{Z}'$ on both sides yields $ER(g_{\mathbf{Z}}) - E(R_{\mathbf{Z}}(g_{\mathbf{Z}})) \leq S(g)$. If $g$ is obtained by *empirical risk minimization* over some hypothesis set $\mathcal{G}$, then $E(R_{\mathbf{Z}}(g_{\mathbf{Z}})) = E(\inf_{g \in \mathcal{G}} R_{\mathbf{Z}}(g)) \leq \inf_{g \in \mathcal{G}} E(R_{\mathbf{Z}}(g)) = \inf_{g \in \mathcal{G}} R(g)$, and one obtains

$$ER(g_{\mathbf{Z}}) - \inf_{g \in \mathcal{G}} R(g) \leq ER(g_{\mathbf{Z}}) - E(R_{\mathbf{Z}}(g_{\mathbf{Z}})) \leq S(g). \quad (3)$$

By eq. (3), the stability defined in (2) yields an upper bound on the generalization error. It can be shown that there exists a converse upper bound, if the minimum is unique and well-separated, such that $ER(g_{\mathbf{Z}}) \to \inf_{g \in \mathcal{G}} R(g)$ implies $S \to 0$.

Note that the stability measures the disagreement between labels on training data and test data, both assigned by $g$. This asymmetry arises naturally and directly measures the generalization performance of $g$. Furthermore, the stability can be interpreted as the expected empirical risk of $g$ with respects to the labels computed by itself (compare (1) and (2)). Therefore, stability measures the *self-consistency* of $g$. This interpretation is also valid in the semi-supervised and unsupervised settings. Practical evaluation of the stability amounts to 2-fold cross-validation. No improvement can therefore be expected in this area. However, unlike cross-validation, stability can also be defined in settings where no label information is available. This property of the method will be discussed in the remainder of this section.

**Semi-supervised Learning**   Semi-supervised learning problems are defined as follows. The label $Y_i$ of an object $X_i$ might not be known. This fact is encoded by setting $Y_i = 0$, since 0 is not a valid label. At least one labeled point must be given for every class. Furthermore, for the present discussion, we assume that we do not have a fully labeled data set for testing purposes.

There exist two alternatives in defining the solution to a semi-supervised learning problem. In the first alternative, the solution is a labeling function $g$ defined on the whole object space $\mathcal{X}$ as in supervised learning. Then, the stability (eq. (2)) can be readily computed and measures the confidence for the (unknown) training error.

The second alternative is that the solution is not given by a labeling function on the whole object space, but only by a labeling function on the training set $\mathbf{Z}$. Labeling functions

which are defined on the training set only will be denoted by $\alpha$ to stress the difference. The labeling on $\mathbf{Z}$ will be denoted by $\alpha_{\mathbf{Z}}$, which is only defined on $\mathbf{X}$. As mentioned above, the stability compares labels given to the training data with predicted labels. In the current setting, there are no predicted labels, because $\alpha$ is defined on the training set only. One possibility to obtain predicted labels is to introduce a predictor $h$, which is trained using $\mathbf{X}, \alpha_{\mathbf{Z}}(\mathbf{X})$ to predict labels on the new set $\mathbf{Z}'$. Leaving $h$ as a free parameter, we define the stability for semi-supervised learning as

$$S_{\text{semi}}^h(\alpha) := E\Big[\frac{1}{n}\sum_{i=1}^n \mathbf{1}\big\{h_{\mathbf{X},\alpha_{\mathbf{Z}}(\mathbf{X})}(X_i') \neq \alpha_{\mathbf{Z}'}(X_i')\big\}\Big]. \tag{4}$$

Of course, the choice of $h$ influences the value of the stability. We need a condition on the prediction step to select $h$. First note that (4) is the expected empirical risk of $h$ with respect to the data source $\mathbf{X}, \alpha_{\mathbf{Z}}(\mathbf{X})$. Analogously to supervised learning, the minimal attainable stability $\min_h S_{\text{semi}}^h(\alpha)$ measures the extent to which classes overlap, or how consistent the labels are. Therefore, $h$ should be chosen to minimize $S_{\text{semi}}^h(\alpha)$. Unfortunately, the construction of non-asymptotically Bayes optimal learning algorithms is extremely difficult and, therefore, we should not expect that there exists a universally applicable constructive procedure for automatically building $h$ given an $\alpha$.

In practice, some $h$ has to be chosen. This choice will yield larger stability costs, i.e. worse stability, and can therefore not fake stability. Furthermore, it is often possible to construct good predictors in practice. Note that (4) measures the mismatch between the label generator $\alpha$ and the predictor $h$. Intuitively, $h$ can lead to good stability only if the strategy of $\alpha$ and $h$ are similar. For unsupervised learning, as discussed in the next paragraph, the choices for various standard techniques are natural. For example, $k$-means clustering suggests to use nearest centroid classification. Minimum spanning tree type clustering algorithms suggest nearest neighbor classifiers, and finally, clustering algorithms which fit a parametric density model should use the class posteriors computed by the Bayes rule for prediction.

**Unsupervised Learning**  The unsupervised learning setting is given as the problem of labeling a finite data set $\mathbf{X} = X_1, \ldots, X_n$. The solution $\alpha_{\mathbf{X}}$ is again a function only defined on $\mathbf{X}$. From this definition, it becomes clear that we again need a predictor as in the second alternative of semi-supervised learning.

For unsupervised learning, another problem arises. Since no specific label values are prescribed for the classes, label indices might be permuted from one instance to another, even when the partitioning is identical. For example, keeping the same classes, exchanging the class labels 1 and 2 leads to a new partitioning, which is not structurally different. In other words, label values are only known up to a permutation. In view of this non-uniqueness of the representation of a partitioning, we define the permutation relating indices on the first set to the second set by the one which maximizes the agreement between the classes. The stability then reads

$$S_{\text{un}}(\alpha) := E\Big[\min_{\pi \in \mathfrak{S}_k} \frac{1}{n}\sum_{i=1}^n \mathbf{1}\big\{\pi(h_{\mathbf{X},\alpha_{\mathbf{X}}(\mathbf{X})}(X_i')) \neq \alpha_{\mathbf{X}'}(X_i')\big\}\Big]. \tag{5}$$

Note that the minimization has to take place inside the expectation, because the permutation depends on the data $\mathbf{X}, \mathbf{X}'$. In practice, it is not necessary to compute all $k!$ permutations, because the problem is solvable by the *Hungarian method* in $O(k^3)$ [11].

**Model Order Selection**  The problem of *model order selection* consists in determining the number of clusters $k$ to be estimated, and exists only in unsupervised learning.

The range of the stability $S$ depends on $k$, therefore stability values cannot be compared for different values of $k$. For unsupervised learning, the stability minimized over $\mathfrak{S}_k$ is bounded from above by $1 - 1/k$, since for a larger instability, there exists a relabeling

which has smaller stability costs. This stability value is asymptotically achieved by the *random predictor* $\rho_k$ which assigns uniformly drawn labels to objects. Normalizing $S$ by the stability of the random predictor yields values independent of $k$. We thus define the re-normalized stability as

$$S_{\text{un}}^k(\alpha) = S_{\text{un}}(\alpha)/S_{\text{un}}(\rho_k). \tag{6}$$

**Resampling Estimate of the Stability**    In practice, a finite data set $X_1, \ldots, X_n$ is given, and the best model should be estimated. The stability is defined in terms of an expectation, which has to be estimated for practical applications. Estimation of $S$ over a hypothesis set $\mathcal{G}$ is feasible if $\mathcal{G}$ has finite VC-dimension, since the VC-dimension for estimating $S$ is the same as for the empirical risk, a fact which is not proved here. In order to estimate the stability, we propose the following resampling scheme: Iteratively split the data set into disjoint halves, and compare the solutions on these sets as defined above for the respective cases. After the model having the smallest value of $S$ is determined, train this model again on the whole data to obtain the result.

Note that it is necessary to split into disjoint subsets, because common points potentially increase the stability artificially. Furthermore, unlike in cross-validation, both sets must have the same size, because both are used as inputs to training algorithms. For semi-supervised and unsupervised learning, the comparison might entail predicting labels on a new set, and for the latter also minimizing over permutation of labels.

## 4   Stability for Model Order Selection in Clustering: Experimental Results

We now provide experimental evidence for the usefulness of our approach to model order selection, which is one of the hardest model assessment problems. First, the algorithms are compared for toy data, in order to study the performance of the stability measure under well-controlled conditions. However, for real-world applications, it does not suffice to be better than competitors, but one has to provide solutions which are reasonable within the framework of the application. Therefore, in a second experiment, the stability measure is compared to the other methods for the problem of clustering gene expression data.

Experiments are conducted using a deterministic annealing variant of $k$-means [12] and Path-Based Clustering [5] optimized via an agglomerative heuristic. For all data sets, we average over 20 resamples for $k = 2, \ldots, 10$. For the Gap Statistic and Clest[1] 20 random samples are drawn from the baseline. For Clest and Prediction Strength, the number of resamples is chosen the same as for our method. The threshold for Prediction Strength is set to 0.9. As mentioned above, the nearest centroid classifier is employed for the purpose of prediction when using $k$-means, and a variant of the nearest neighbor classifier is used for Path-Based Clustering which can be regarded as a combination of Minimum Spanning Tree clustering and Pairwise Clustering [5, 8].

We compare the proposed stability index of section 3 with the Gap Statistic, Clest and with Tibshirani's Prediction Strength method using two toy data sets and a microarray data set taken from [7]. Table 1 summarizes the estimated number of clusters $\hat{k}$ of each method.

**Toy Data Sets**    The first data set consists of three fairly well separated point clouds, generated from three Gaussian distributions (25 points from the first and the second and 50 points from the third were drawn). Note that for some $k$, for example $k = 5$ in figure 1(a), the variance in the stability over different resamples is quite high. This effect is due to the model mismatch, since for $k = 5$, the clustering of the three classes depends highly on the subset selected in the resampling. This means that besides the absolute value of the stability

| Data Set | Stability Method | Gap Statistic | Clest | Prediction Strength | "true" number $k$ |
|---|---|---|---|---|---|
| 3 Gaussians | $\hat{k} = 3$ | $\hat{k} = 3$ | $\hat{k} = 3$ | $\hat{k} = 3$ | $k = 3$ |
| 3 Rings $k$-means | $\hat{k} = 7$ | $\hat{k} = 1$ | $\hat{k} = 7$ | $\hat{k} = 1$ | $k = 3$ |
| 3 Rings Path-Based | $\hat{k} = 3$ | $\hat{k} = 1$ | $\hat{k} = 1$ | $\hat{k} = 1$ | $k = 3$ |
| Golub *et al.* data | $\hat{k} = 3$ | $\hat{k} = 10$ | $\hat{k} = 3$ | $\hat{k} = 1$ | $k = 3$ or $k = 2$ |

Table 1: The estimated model orders for the two toy and the microarray data set.

costs, additional information about the fit can be obtained from the distribution of the stability costs over the resampled subsets. For this data set, all methods under comparison are able to infer the "true" number of clusters $k = 3$. Figures 1(d) and 1(a) show the clustered data set and the proposed stability index. For $k = 2$, the stability is relatively high, which is due to the hierarchical structure of the data set, which enables stable merging of the two smaller sub-clusters.

In the ring data set (depicted in figures 1(e) and 1(f)), one can naturally distinguish three ring shaped clusters that violate the modeling assumptions of $k$-means since clusters are not spherically distributed. Here, $k$-means is able to identify the inner circle as a cluster with $k = 7$. Thus, the stability for this number of clusters $k$ is highest (figure 1(b)). All other methods except Clest infer $\hat{k} = 1$ for this data set with $k$-means. Applying the proposed stability estimator with Path-Based Clustering on the same data set yields highest stability for $k = 3$, the "correct" number of clusters (figures 1(f) and 1(c)). Here, all other methods fail and estimate $\hat{k} = 1$. The Gap Statistic fails here because it directly incorporates the assumption of spherically distributed data. Similarly, the Prediction Strength measure and Clest (in the form we use here) use classifiers that only support linear decision boundaries which obviously cannot discriminate between the three ring-shaped clusters. In all these cases, the basic requirement for a validation scheme is violated, namely that it must not incorporate additional assumptions about the group structure in a data set that go beyond the ones of the clustering principle employed. Apart from that, it is noteworthy that the stability with $k$-means is significantly worse than the one achieved with Path-Based Clustering, which indicates that the latter is the better choice for this data set.

**Application to Microarray Data**   Recently, several authors have investigated the possibility of identifying novel tumor classes based solely on gene expression data [7, 2, 1]. Golub et al. [7] studied in their analysis the problem of classifying and clustering acute leukemias. The important question of inferring an appropriate model order remains unaddressed in their article and prior knowledge is used instead. In practice however, such knowledge is often not available.

Acute leukemias can be roughly divided into two groups, *acute myeloid leukemia (AML)* and *acute lymphoblastic leukemia (ALL)* where the latter can furthermore be subdivided into B-cell ALL and T-cell ALL. Golub et al. used a data set of 72 leukemia samples (25 AML, 47 ALL of which 38 are B-cell ALL samples)[2]. For each sample, gene expression was monitored using Affymetrix expression arrays.

We apply the preprocessing steps as in Golub et al. resulting in a data set consisting of 3571 genes and 72 samples. For the purpose of cluster analysis, the feature set was additionally reduced by only retaining the 100 genes with highest variance across samples. This step is adopted from [6]. The final data set consists of 100 genes and 72 samples. We have performed cluster analysis using $k$-means and the nearest centroid rule. Figure 2 shows

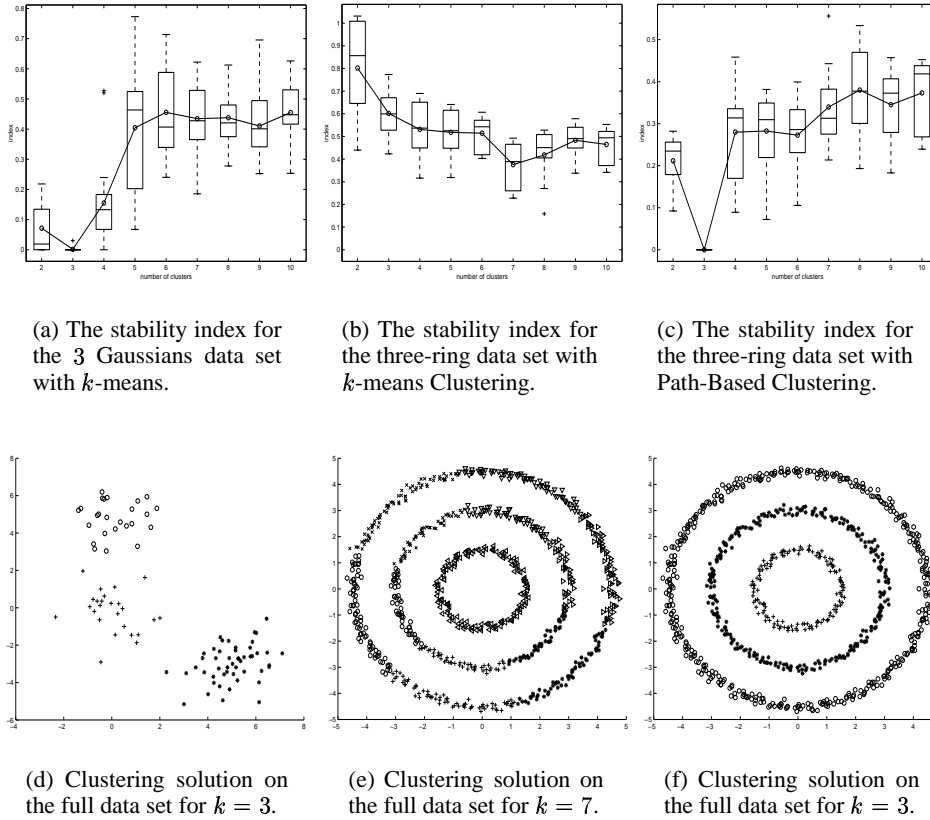

(a) The stability index for the 3 Gaussians data set with $k$-means.

(b) The stability index for the three-ring data set with $k$-means Clustering.

(c) The stability index for the three-ring data set with Path-Based Clustering.

(d) Clustering solution on the full data set for $k = 3$.

(e) Clustering solution on the full data set for $k = 7$.

(f) Clustering solution on the full data set for $k = 3$.

Figure 1: Results of the stability index on the toy data (see section 4).

the corresponding stability curve. For $k = 3$, we estimate the highest stability. We expect that clustering with $k = 3$ separates AML, B-cell ALL and T-cell ALL samples from each other. With respect to the known ground-truth labels, $91.6\%$ of the samples (66 samples) are correctly classified (the Hungarian method is used to map the clusters to the ground-truth). Of the competitors, only Clest is able to infer the "correct" number of cluster $k = 3$ while the Gap Statistic largely overestimates the number of clusters. The Prediction strength does not provide any reasonable result as it estimates $\hat{k} = 1$. Note, that for $k = 2$ similar stability is achieved. We cluster the data set again for $k = 2$ and compare the result with the ALL – AML labeling of the data. Here, $86.1\%$ of the samples (62 samples) are correctly identified. We conclude that our method is able to infer biologically relevant model orders. At the same time, a $k$ is suggested that leads to high accuracy w.r.t. the ground-truth. Hence, our re-analysis demonstrates that we could have recovered a biologically meaningful grouping in a completely unsupervised manner.

## 5   Conclusion

The problem of model assessment was addressed in this paper. The goal was to derive a common framework for practical assessment of learning models. Starting with defining a stability measure in the context of supervised learning, this measure was generalized to semi-supervised and unsupervised learning. The experiments concentrated on model or-

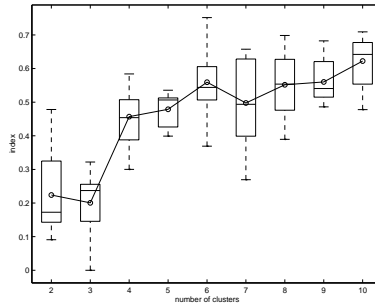

Figure 2: Resampled stability for the leukemia dataset vs. number of classes (see sec. 4).

der selection for unsupervised learning, because this is the area where the need for widely applicable model assessment strategies is highest. On toy data, the stability measure outperforms other techniques, when their respective modeling assumptions are violated. On real-world data, the stability measure compares favorably to the best of the competitors.

**Acknowledgments.** This work has been supported by the German Research Foundation (DFG), grants #Buh 914/4, #Buh 914/5.

## Footnotes

[1]See section 2 for a brief overview over these techniques.

[2]Available at `http://www-genome.wi.mit.edu/cancer/`

# References

[1] A. A. Alizadeh et al. Distinct types of diffuse large b-cell lymphoma identified by gene expression profiling. *Nature*, 403:503 – 511, 2000.

[2] M. Bittner et al. Molecular classification of cutaneous malignant melanoma by gene expression profiling. *Nature*, 406(3):536 – 540, 2000.

[3] O. Bousquet and A. Elisseeff. Stability and generalization. *Journal of Machine Learning Research*, 2:499–526, 2002.

[4] J. Breckenridge. Replicating cluster analysis: Method, consistency and validity. *Multivariate Behavioural research*, 1989.

[5] B. Fischer, T. Zöller, and J. M. Buhmann. Path based pairwise data clustering with application to texture segmentation. In *LNCS Energy Minimization Methods in Computer Vision and Pattern Recognition*. Springer Verlag, 2001.

[6] J. Fridlyand and S. Dudoit. Applications of resampling methods to estimate the number of clusters and to improve the accuracy of a clustering method. Technical Report 600, Statistics Department, UC Berkeley, September 2001.

[7] T.R. Golub et al. Molecular classification of cancer: Class discovery and class prediction by gene expression monitoring. *Science*, pages 531 – 537, October 1999.

[8] T. Hofmann and J. M. Buhmann. Pairwise data clustering by deterministic annealing. *IEEE PAMI*, 19(1), January 1997.

[9] A. K. Jain and R. C. Dubes. *Algorithms for Clustering Data*. Prentice-Hall, Inc., 1988.

[10] Michael J. Kearns and Dana Ron. Algorithmic stability and sanity-check bounds for leave-one-out cross-validation. In *Computational Learing Theory*, pages 152–162, 1997.

[11] H.W. Kuhn. The hungarian method for the assignment problem. *Naval Res. Logist. Quart.*, 2:83–97, 1955.

[12] K. Rose, E. Gurewitz, and G. C. Fox. A deterministic annealing approach to clustering. *Pattern Recognition Letters*, 11(9):589 – 594, 1990.

[13] R. Tibshirani, G. Walther, D. Botstein, and P. Brown. Cluster validation by prediction strength. Technical report, Statistics Department, Stanford University, September 2001.

[14] R. Tibshirani, G. Walther, and T. Hastie. Estimating the number of clusters via the gap statistic. Technical report, Statistics Department, Stanford University, March 2000.